# Fast Non-Linear Dimension Reduction

**Nanda Kambhatla and Todd K. Leen**
Department of Computer Science and Engineering
Oregon Graduate Institute of Science & Technology
P.O. Box 91000 Portland, OR 97291-1000

## Abstract

We present a fast algorithm for non-linear dimension reduction. The algorithm builds a _local linear_ model of the data by merging PCA with clustering based on a new distortion measure. Experiments with speech and image data indicate that the local linear algorithm produces encodings with lower distortion than those built by five layer auto-associative networks. The local linear algorithm is also more than an order of magnitude faster to train.

## 1 Introduction

Feature sets can be more compact than the data they represent. Dimension reduction provides compact representations for storage, transmission, and classification. Dimension reduction algorithms operate by identifying and eliminating statistical redundancies in the data.

The optimal linear technique for dimension reduction is principal component analysis (PCA). PCA performs dimension reduction by projecting the original $n$-dimensional data onto the $m < n$ dimensional linear subspace spanned by the leading eigenvectors of the data's covariance matrix. Thus PCA builds a _global linear_ model of the data (an $m$ dimensional hyperplane). Since PCA is sensitive only to correlations, it fails to detect higher-order statistical redundancies. One expects non-linear techniques to provide better performance; i.e. more compact representations with lower distortion.

This paper introduces a _local linear_ technique for non-linear dimension reduction. We demonstrate its superiority to a recently proposed global non-linear technique,

and show that both non-linear algorithms provide better performance than PCA for speech and image data.

## 2    Global Non-Linear Dimension Reduction

Several researchers (e.g. Cottrell and Metcalfe 1991) have used layered feedforward auto-associative networks with a bottle-neck middle layer to perform dimension reduction. It is well known that auto-associative nets with a single hidden layer cannot provide lower distortion than PCA (Bourlard and Kamp, 1988). Recent work (e.g. Oja 1991) shows that *five layer* auto-associative networks *can* improve on PCA. These networks have three hidden layers (see Figure 1(a)). The first and third hidden layers have non-linear response, and are referred to as the *mapping layers*. The $m < n$ nodes of the middle or *representation layer* provide the encoded signal.

The first two layers of weights produce a projection from $\mathcal{R}^n$ to $\mathcal{R}^m$. The last two layers of weights produce an immersion from $\mathcal{R}^m$ into $\mathcal{R}^n$. If these two maps are well chosen, then the complete mapping from input to output will approximate the identity for the training data. If the data requires the projection and immersion to be non-linear to achieve a good fit, then the network can in principal find such functions.

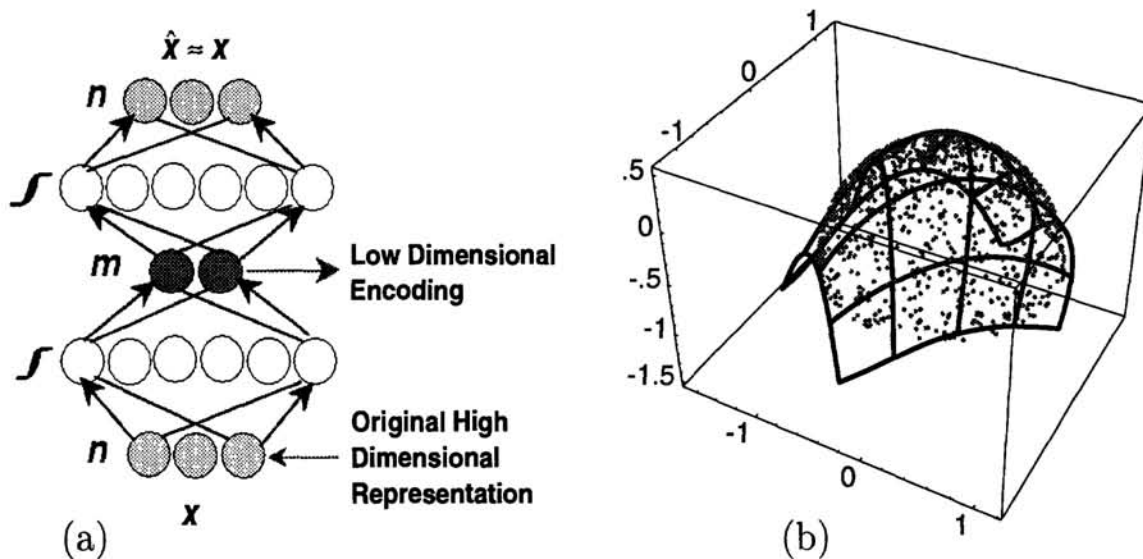

Figure 1: (a) A five layer feedforward auto-associative network. This network can perform a non-linear dimension reduction from $n$ to $m$ dimensions. (b) Global curvilinear coordinates built by a five layer network for data distributed on the surface of a hemisphere. When the activations of the representation layer are swept, the outputs trace out the curvilinear coordinates shown by the solid lines.

The activities of the nodes in the representation layer form global curvilinear co-ordinates on a submanifold of the input space (see Figure 1(b)). We thus refer to five layer auto-associative networks as a *global, nonlinear* dimension reduction technique.

## 3 Locally Linear Dimension Reduction

Five layer networks have drawbacks; they can be very slow to train and they are prone to becoming trapped in poor local optima. Furthermore, it may not be possible to accurately fit global, low dimensional, curvilinear coordinates to the data. We propose an alternative that does not suffer from these problems.

Our algorithm pieces together local linear coordinate patches. The local regions are defined by the partition of the input space induced by a vector quantizer (VQ). The orientation of the local coordinates is determined by PCA (see Figure 2). In this section, we present two ways to obtain the partition. First we describe an approach that uses Euclidean distance, then we describe a new distortion measure which is optimal for our task (local PCA).

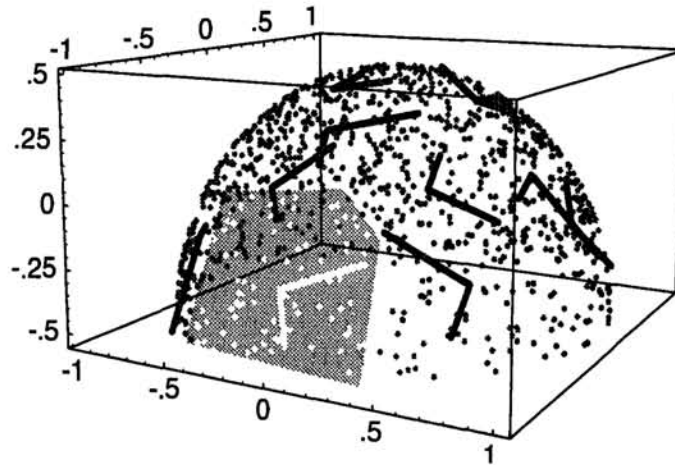

Figure 2: Local coordinates built by our algorithm (dubbed VQPCA) for data distributed on the surface of a hemisphere. The solid lines represent the two principal eigen-directions in each Voronoi cell. The region covered by one Voronoi cell is shown shaded.

### 3.1 Euclidean partitioning

Here, we do a clustering (with Euclidean distance) followed by PCA in each of the local regions. The hybrid algorithm, dubbed VQPCA, proceeds in three steps:

1. Using competitive learning, train a VQ (with Euclidean distance) with $Q$ reference vectors (weights) $(r_1, r_2, \ldots, r_Q)$.

2. Perform a *local* PCA within each Voronoi cell of the VQ. For each cell, compute the local covariance matrix for the data with respect to the corresponding reference vector (centroid) $r_c$. Next compute the eigenvectors $(e_1^c, \ldots, e_n^c)$ of each covariance matrix.

3. Choose a target dimension $m$ and project each data vector $x$ onto the leading $m$ eigenvectors to obtain the local linear coordinates $z = (e_1^c \cdot (x - r_c), \ldots, e_m^c \cdot (x - r_c))$.

The encoding of $x$ consists of the index $c$ of the reference cell closest (Euclidean distance) to $x$, together with the $m < n$ component vector $z$. The decoding is given by

$$\hat{x} = r_c + \sum_{i=1}^{m} z_i e_i^c \,, \qquad (1)$$

where $r_c$ is the *reference vector* (centroid) for the cell $c$, and $e_i^c$ are the leading eigenvectors of the covariance matrix of the cell $c$. The mean squared reconstruction error incurred by VQPCA is

$$\mathcal{E}_{recon} = E[\, \|x - \hat{x}\|^2 \,] = E[\, \|x - r_c - \sum_{i=1}^{m} z_i e_i^c\|^2 \,] \qquad (2)$$

where $E[\cdot]$ denotes an expectation with respect to $x$, and $\hat{x}$ is defined in (1).

Training the VQ and performing the local PCA are very fast relative to training a five layer network. The training time is dominated by the distance computations for the competitive learning. This computation can be speeded up significantly by using a multi-stage architecture for the VQ (Gray 1984).

## 3.2   Projection partitioning

The VQPCA algorithm as described above is not optimal because the clustering is done independently of the PCA projection. The goal is to minimize the expected error in reconstruction (2). We can realize this by using the expected reconstruction error as the distortion measure for the design of the VQ.

The reconstruction error for VQPCA ($\mathcal{E}_{recon}$ defined in (2)) can be written in matrix form as

$$\mathcal{E}_{recon} = E[\, (x - r_c)^T P_c^T P_c (x - r_c)] \,, \qquad (3)$$

where $P_c$ is an $m \times n$ matrix whose rows are the orthonormal trailing eigenvectors of the covariance matrix for the cell $c$. This is the mean squared Euclidean distance between the data and the local hyperplane.

The expression for the VQPCA error in (2) suggests the distortion measure

$$d(x, r_c) = (x - r_c)^T P_c^T P_c (x - r_c) \,. \qquad (4)$$

We call this the *reconstruction distance*. The reconstruction distance is the error incurred in approximating $x$ using *only* $m$ local PCA coefficients. It is the squared projection of the difference vector $x - r_c$ on the *trailing* eigenvectors of the covariance matrix for the cell $c$. Clustering with respect to the reconstruction distance directly minimizes the expected reconstruction error $\mathcal{E}_{recon}$.

The modified VQPCA algorithm is:

1. Partition the input space using a VQ with the reconstruction distance measure [1] in (4).

2. Perform a local PCA (same as in steps 2 and 3 of the algorithm as described in section 3.1).

# 4   Experimental Results

We apply PCA, five layer networks (5LNs), and VQPCA to dimension reduction of speech and images. We compare the algorithms using two performance criteria: training time and the distortion in the reconstructed signal. The distortion measure is the normalized reconstruction error:

$$\mathcal{E}_{norm} \;=\; \frac{\mathcal{E}_{recon}}{E[\,\|x\|^2\,]} \;=\; \frac{E[\,\|\,x - \hat{x}\,\|^2\,]}{E[\,\|x\|^2\,]}\,.$$

## 4.1   Model Construction

The 5LNs were trained using three optimization techniques: conjugate gradient descent (CGD), the BFGS algorithm (a quasi-Newton method (Press *et al* 1987)), and stochastic gradient descent (SGD). In order to limit the space of architectures, the 5LNs have the same number of nodes in both of the mapping (second and fourth) layers.

For the VQPCA with Euclidean distance, clustering was implemented using standard VQ (VQPCA-Eucl) and multistage quantization (VQPCA-MS-E). The multistage architecture reduces the number of distance calculations and hence the training time for VQPCA (Gray 1984).

## 4.2   Dimension Reduction of Speech

We used examples of the twelve monothongal vowels extracted from continuous speech drawn from the TIMIT database (Fisher and Doddington 1986). Each input vector consists of 32 DFT coefficients (spanning the frequency range 0-4kHz), time-averaged over the central third of the utterance. We divided the data set into a training set containing 1200 vectors, a validation set containing 408 vectors and a test set containing 408 vectors. The validation set was used for architecture selection (e.g the number of nodes in the mapping layers for the five layer nets). The test set utterances are from speakers *not* represented in the training set or the validation set. Motivated by the desire to capture formant structure in the vowel encodings, we reduced the data from 32 to 2 dimensions. (Experiments on reduction to 3 dimensions gave similar results to those reported here (Kambhatla and Leen 1993).)

Table 1 gives the **test set** reconstruction errors and the training times. The VQPCA encodings have significantly lower reconstruction error than the global PCA or five layer nets. The best 5LNs have slightly lower reconstruction error than PCA, but are very slow to train. Using the multistage search, VQPCA trains more than two orders of magnitude faster than the best 5LN, and achieves an error about 0.7 times as great. The modified VQPCA algorithm (with the reconstruction distance measure used for clustering) provides the least reconstruction error among all the architectures tried.

Table 1: Speech data **test set** reconstruction errors and training times. Architectures represented here are from experiments with the lowest validation set error over the parameter ranges explored. The numbers in the parentheses are the values of the free parameters for the algorithm represented (e.g 5LN-CGD (5) indicates a network with 5 nodes in both the mapping (2nd and 4th) layers, while VQPCA-Eucl (50) indicates a clustering into 50 Voronoi cells).

| ALGORITHM | $\mathcal{E}_{norm}$ | TRAINING TIME (in seconds) |
|---|---|---|
| PCA | 0.0060 | 11 |
| 5LN-CGD (5) | 0.0069 | 956 |
| 5LN-BFGS (30) | 0.0057 | 28,391 |
| 5LN-SGD (25) | 0.0055 | 94,903 |
| VQPCA-Eucl (50) | 0.0037 | 1,454 |
| VQPCA-MS-E (9x9) | 0.0036 | 142 |
| VQPCA-Recon (45) | 0.0031 | 931 |

Table 2: Reconstruction errors and training times for a 50 to 5 dimension reduction of images. Architectures represented here are from experiments with the lowest validation set error over the parameter ranges explored.

| ALGORITHM | $\mathcal{E}_{norm}$ | TRAINING TIME (in seconds) |
|---|---|---|
| PCA | 0.458 | 5 |
| 5LN-CGD (40) | 0.298 | 3,141 |
| 5LN-BFGS (20) | 0.052 | 10,389 |
| 5LN-SGD (25) | 0.350 | 15,486 |
| VQPCA-Eucl (20) | 0.140 | 163 |
| VQPCA-MS-E (8x8) | 0.176 | 118 |
| VQPCA-Recon (25) | 0.099 | 108 |

## 4.3   Dimension Reduction of Images

The data consists of 160 images of the faces of 20 people. Each is a 64x64, 8-bit/pixel grayscale image. We extracted the first 50 principal components of each image and use these as our experimental data. This is the same data and preparation that DeMers and Cottrell used in their study of dimension reduction with five layer auto-associative nets (DeMers and Cottrell 1993). They trained auto-associators to reduce the 50 principal components to 5 dimensions.

We divided the data into a training set containing 120 images, a validation set (for architecture selection) containing 20 images and a test set containing 20 images. We reduced the images to 5 dimensions using PCA, 5LNs[2] and VQPCA. Table 2

Table 3: Reconstruction errors and training times for a 50 to 5 dimension reduction of images (training with *all* the data). Architectures represented here are from experiments with the lowest error over the parameter ranges explored.

| ALGORITHM | $\mathcal{E}_{\mathbf{norm}}$ | TRAINING TIME (in seconds) |
|---|---|---|
| PCA | 0.4054 | 7 |
| 5LN-SGD (30) | 0.1034 | 25,306 |
| 5LN-SGD (40) | 0.0729 | 31,980 |
| VQPCA-Eucl (50) | 0.0009 | 905 |
| VQPCA-Recon (50) | 0.0017 | 216 |

summarizes the results. We notice that a five layer net obtains the encoding with the least error for this data, but it takes a long time to train. Presumably more training data would improve the best VQPCA results.

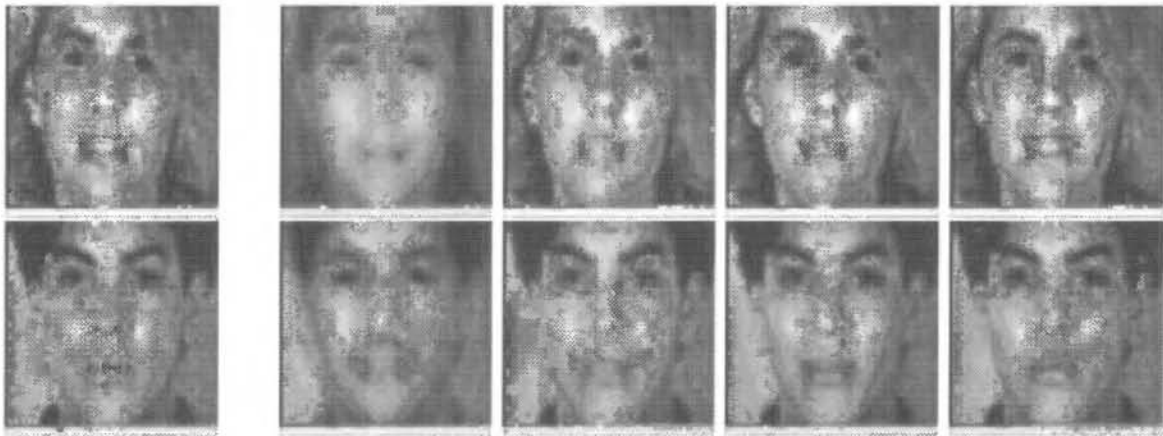

Figure 3: Two representative images: Left to right – Original 50-PC image, reconstruction from 5-D encodings: PCA, 5LN-SGD(40), VQPCA(10), and VQPCA(50).

For comparison with DeMers and Cottrell's (DeMers and Cottrell 1993) work, we also conducted experiments training with *all* the data. The results are summarized[3] in Table 3 and Figure 3 shows two sample faces. Both non-linear techniques produce encodings with lower error than PCA, indicating significant non-linear structure in the data. With the same data, and with a 5LN with 30 nodes in each mapping layer, DeMers (DeMers and Cottrell 1993) obtains a reconstruction error $\mathcal{E}_{norm}$ 0.1317[4]. We note that the VQPCA algorithms achieve an order of magnitude improvement over five layer nets *both* in terms of speed of training and the accuracy of encodings.

## 5 Summary

We have presented a local linear algorithm for dimension reduction. We propose a new distance measure which is optimal for the task of local PCA. Our results with speech and image data indicate that the nonlinear techniques provide more accurate encodings than PCA. Our local linear algorithm produces more accurate encodings (except for one simulation with image data), and trains much faster than five layer auto-associative networks.

**Acknowledgments**

This work was supported by grants from the Air Force Office of Scientific Research (F49620-93-1-0253) and Electric Power Research Institute (RP8015-2). The authors are grateful to Gary Cottrell and David DeMers for providing their image database and clarifying their experimental results. We also thank our colleagues in the Center for Spoken Language Understanding at OGI for providing speech data.

**References**

H. Bourlard and Y. Kamp. (1988) Auto-association by multilayer perceptrons and singular value decomposition. *Biological Cybernetics*, 59:291-294.

G. Cottrell and J. Metcalfe. (1991) EMPATH: Face, emotion, and gender recognition using holons. In R. Lippmann, John Moody and D. Touretzky, editors, *Advances in Neural Information Processing Systems 3*, pages 564-571. Morgan Kauffmann.

D. DeMers and G. Cottrell. (1993) Non-linear dimensionality reduction. In Giles, Hanson, and Cowan, editors, *Advances in Neural Information Processing Systems 5*. San Mateo, CA: Morgan Kaufmann.

W. M. Fisher and G. R. Doddington. (1986) The DARPA speech recognition research database: specification and status. In *Proceedings of the DARPA Speech Recognition Workshop*, pages 93-99, Palo Alto, CA.

A. Gersho and R. M. Gray. (1992) Vector Quantization and Signal Compression. Kluwer academic publishers.

R. M. Gray. (1984) Vector quantization. *IEEE ASSP Magazine*, pages 4-29.

N. Kambhatla and T. K. Leen. (1993) Fast non-linear dimension reduction. In *IEEE International Conference on Neural Networks*, Vol. 3, pages 1213-1218. IEEE.

E. Oja. (1991) Data compression, feature extraction, and autoassociation in feed-forward neural networks. In *Artificial Neural Networks*, pages 737-745. Elsevier Science Publishers B.V. (North-Holland).

W. H. Press, B. P. Flannery, S. A. Teukolsky, and W. T. Vetterling. (1987) *Numerical Recipes - the Art of Scientific Computing*. Cambridge University Press, Cambridge/New York.

## Footnotes

[1]The VQ is trained using the (batch mode) generalized Lloyd's algorithm (Gersho and Gray, 1992) rather than an on-line competitive learning. This avoids recomputing the matrix $P_c$ (which depends on $r_c$) for each input vector.

[2]We used 5LNs with a configuration of 50-$n$-5-$n$-50, $n$ varying from 10 to 40 in increments of 5. The BFGS algorithm posed prohibitive memory and time requirements for $n > 20$ for this task.

[3]For 5LNs, we only show results with SGD in order to compare with the experimental results of DeMers. For this data, 5LN-CGD gave encodings with a higher error and 5LN-BFGS posed prohibitive memory and computational requirements.

[4]DeMers reports half the MSE per output node, $E = (1/2) * (1/50) * MSE = 0.001$. This corresponds to $\mathcal{E}_{norm} = 0.1317$
